# A Probabilistic Approach to Single Channel Blind Signal Separation

**Gil-Jin Jang**
Spoken Language Laboratory
KAIST, Daejon 305-701, South Korea
*jangbal@bawi.org*
*http://speech.kaist.ac.kr/~jangbal*

**Te-Won Lee**
Institute for Neural Computation
University of California, San Diego
La Jolla, CA 92093, U.S.A.
*tewon@inc.ucsd.edu*

## Abstract

We present a new technique for achieving source separation when given only a single channel recording. The main idea is based on exploiting the inherent time structure of sound sources by learning *a priori* sets of basis filters in time domain that encode the sources in a statistically efficient manner. We derive a learning algorithm using a maximum likelihood approach given the observed single channel data and sets of basis filters. For each time point we infer the source signals and their contribution factors. This inference is possible due to the prior knowledge of the basis filters and the associated coefficient densities. A flexible model for density estimation allows accurate modeling of the observation and our experimental results exhibit a high level of separation performance for mixtures of two music signals as well as the separation of two voice signals.

## 1 Introduction

Extracting individual sound sources from an additive mixture of different signals has been attractive to many researchers in computational auditory scene analysis (CASA) [1] and independent component analysis (ICA) [2]. In order to formulate the problem, we assume that the observed signal $y^t$ is an addition of $P$ independent source signals

$$y^t = \lambda_1 x_1^t + \lambda_2 x_2^t + \ldots + \lambda_P x_P^t \,, \tag{1}$$

where $x_i^t$ is the $t^{\text{th}}$ sampled value of the $i^{\text{th}}$ source signal, and $\lambda_i$ is the gain of each source which is fixed over time. Note that superscripts indicate sample indices of time-varying signals and subscripts indicate the source identification. The gain constants are affected by several factors, such as powers, locations, directions and many other characteristics of the source generators as well as sensitivities of the sensors. It is convenient to assume all the sources to have zero mean and unit variance. The goal is to recover all $x_i^t$ given only a single sensor input $y^t$. The problem is too ill-conditioned to be mathematically tractable since the number of unknowns is $PT + P$ given only $T$ observations. Several earlier attempts [3, 4, 5, 6] to this problem have been proposed based on the presumed properties of the individual sounds in the frequency domain.

ICA is a data driven method which relaxes the strong characteristical frequency structure assumptions. However, ICA algorithms perform best when the number of the observed

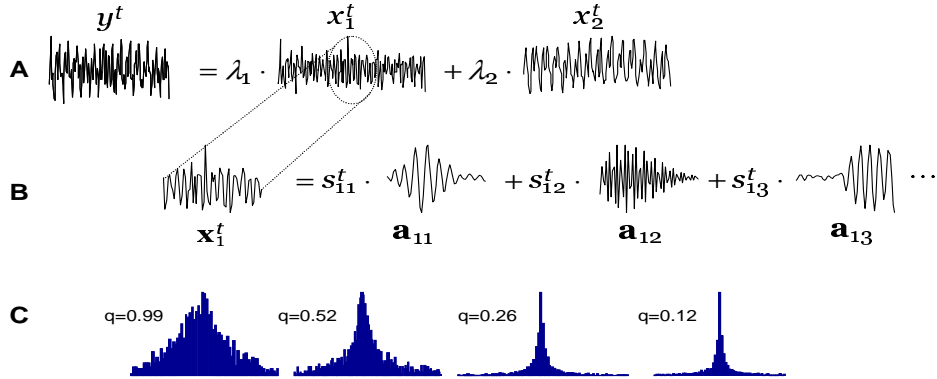

Figure 1: Generative models for the observed mixture and original source signals (**A**) A single channel observation is generated by a weighted sum of two source signals with different characteristics. (**B**) Individual source signals are generated by weighted ($s_{ik}^t$) linear superpositions of basis functions ($\mathbf{a}_{ik}$). (**C**) Examples of the actual coefficient distributions. They generally have more sharpened summits and longer tails than a Gaussian distribution, and would be classified as super-Gaussian. The distributions are modeled by generalized Gaussian density functions in the form of $p(s_{ik}^t) \propto \exp\left(-|s_{ik}^t|^q\right)$, which provide good matches to the non-Gaussian distributions by varying exponents. From left to right, the exponent decreases, and the distribution becomes more super-Gaussian.

signals is greater than or equal to the number of sources [2]. Although some recent overcomplete representations may relax this assumption, the problem of separating sources from a single channel observation remains difficult. ICA has been shown to be highly effective in other aspects such as encoding speech signals [7] and natural sounds [8]. The basis functions and the coefficients learned by ICA constitute an efficient representation of the given time-ordered sequences of a sound source by estimating the maximum likelihood densities, thus reflecting the statistical structures of the sources.

The method presented in this paper aims at exploiting the ICA basis functions for separating mixed sources from a single channel observation. Sets of basis functions are learned a priori from a training data set and these sets are used to separate the unknown test sound sources. The algorithm recovers the original auditory streams in a number of gradient-ascent adaptation steps maximizing the log-likelihood of the separated signals, calculated using the basis functions and the probability density functions (pdf's) of their coefficients —the output of the ICA basis filters. The object function not only makes use of the ICA basis functions as a strong prior for the source characteristics, but also their associated coefficient pdf's modeled by generalized Gaussian distributions [9]. Experiments showing the separation of the two different sources was quite successful in the simulated mixtures of rock and jazz music, and male and female speech signals.

## 2 Generative Models for Mixture and Source Signals

The algorithm first involves the learning of the time-domain basis functions of the sound sources that we are interested in separating from a given training database. This corresponds to the prior information necessary to successfully separate the signals. We assume two different types of generative models in the observed single channel mixture as well as in the original sources. The first one is depicted in Figure 1-**A**. As described in Equation 1, at every $t \in [1, T]$ the observed instance is assumed to be a weighted sum of different sources. In our approach only the case of $P = 2$ is regarded. This corresponds to the situ-

ation defined in Section 1 in that two different signals are mixed and observed in a single sensor.

For the individual source signals, we adopt a decomposition-based approach as another generative model. This approach was employed formerly in analyzing sound sources [7, 8] by expressing a fixed-length segment drawn from a time-varying signal as a linear super-position of a number of elementary patterns, called basis functions, with scalar multiples (Figure 1-**B**). Continuous samples of length $N$ with $N \ll T$ are chopped out of a source, from $t$ to $t + N - 1$, and the subsequent segment is denoted as an $N$-dimensional column vector in a boldface letter, $\mathbf{x}_i^t = [x_i^t \ x_i^{t+1} \ \ldots \ x_i^{t+N-1}]'$, attaching the lead-off sample index for the superscript and representing the transpose operator with $'$. The constructed column vector is then expressed as a linear combination of the basis functions such that

$$\mathbf{x}_i^t = \sum_{k=1}^{M} \mathbf{a}_{ik} s_{ik}^t = \mathbf{A}_i \mathbf{s}_i^t, \tag{2}$$

where $M$ is the number of basis functions, $\mathbf{a}_{ik}$ is the $k^{\text{th}}$ basis function of $i^{\text{th}}$ source in the form of $N$-dimensional column vector, $s_{ik}^t$ its coefficient (weight) and $\mathbf{s}_i^t = [s_{i1}^t \ s_{i2}^t \ldots s_{iM}^t]'$. The r.h.s. is the matrix-vector notation. The second subscript $k$ followed by the source index $i$ in $s_{ik}^t$ represents the component number of the coefficient vector $\mathbf{s}_i^t$. We assume that $M = N$ and $\mathbf{A}$ has full rank so that the transforms between $\mathbf{x}_i^t$ and $\mathbf{s}_i^t$ be reversible in both directions. The inverse of the basis matrix, $\mathbf{W}_i = \mathbf{A}_i^{-1}$, refers to the ICA filters that generate the coefficient vector: $\mathbf{s}_i^t = \mathbf{W}_i \mathbf{x}_i^t$. The purpose of this decomposition is to model the multivariate distribution of $\mathbf{x}_i^t$ in a statistically efficient manner. The ICA learning algorithm is equivalent to searching for the linear transformation that make the components as statistically independent as possible, as well as maximizing the marginal densities of the transformed coordinates for the given training data [10],

$$\mathbf{W}_i^* = \arg \max_{\mathbf{W}_i} \prod_t \Pr(\mathbf{x}_i^t | \mathbf{W}_i) = \arg \max_{\mathbf{W}_i} \prod_t \prod_k \Pr(s_{ik}^t), \tag{3}$$

where $\Pr(a)$ denotes the probability of the value of a variable $a$. Independence between the components and over time samples factorizes the joint probabilities of the coefficients into the product of marginal ones. What matters is therefore how well matched the model distribution is to the true underlying distribution of $\Pr(s_{ik}^t)$. The coefficient histogram of real data reveals that the distribution has a highly sharpened point at the peak with a long tail (Figure 1-**C**). Therefore we use a generalized Gaussian prior [9] that provides an accurate estimate for symmetric non-Gaussian distributions by fitting the exponent $q$ in the set of parameters $\theta$ in its simplest form

$$p(s|\theta) \propto \exp\left[-\left|\frac{s-\mu}{\sigma}\right|^q\right], \quad \theta = \{\mu, \sigma, q\} \tag{4}$$

where $\mu = E[s]$, $\sigma = \sqrt{V[s]}$, and $p(a)$ is a realized pdf of variable $a$ and should be noted distinctively with $\Pr(a)$. With the generalized Gaussian ICA learning algorithm [9], the basis functions and their individual parameter set $\theta_{ik}$ are obtained beforehand and used as prior information for the following source separation algorithm.

## 3    Separation Algorithm

The method is motivated by the pdf approximation property of ICA transformation (Equation 3). The probability of the source signals is computed by the generalized Gaussian parameters in the transformed domain, and the method performs *maximum a posteriori* (MAP) estimation in a number of adaptation steps on the source signals to maximize the data likelihood. Scaling factors of the generative model are learned as well.

### 3.1 MAP estimation of Source Signals

We have demonstrated that the learned basis filters maximize the likelihood of the given data. Suppose we know what kind of sound sources have been mixed and we were given the set of basis filters from a training set. Could we infer the learning data? The answer is generally "no" when $N < T$ and no other information is given. In our problem of single channel separation, half of the solution is already given by the constraint $y^t = \lambda_1 x_1^t + \lambda_2 x_2^t$, where $x_i^t$ constitutes the basis learning data $\mathbf{x}_i^t$ (Figure 1-**B**). Essentially, the goal of the source inferring algorithm presented in this paper is to complement the remaining half with the statistical information given by a set of coefficient density parameters $\theta_{ik}$. If model parameters are given, we can perform *maximum a posteriori* (MAP) estimation simply by optimizing the data likelihood computed by the model parameters.

At every time point a segment $\mathbf{x}_1^t = [x_1^t \ldots x_1^{t+N-1}]'$ generates the independent coefficient vector $\mathbf{s}_1^t = \mathbf{W}_1 \mathbf{x}_1^t$ and $\mathbf{s}_2^t = \mathbf{W}_2 \mathbf{x}_2^t$ respectively. The likelihood of $\mathbf{x}_1^t$ is

$$\Pr(\mathbf{x}_1^t | \mathbf{W}_1) \cong p(\mathbf{s}_1^t | \Theta_1) | \det \mathbf{W}_1 |, \tag{5}$$

where $p(\cdot)$ is the generalized Gaussian density function, and $\Theta_1 = \theta_{1,1\ldots M}$ — parameter group of all the coefficients, with the notation '$i \ldots j$' meaning an ordered set of the elements from index $i$ to $j$. Assuming the independence over time, the probability of the whole signal $x_1^{1\ldots T}$ is obtained from the marginal ones of all the possible segments,

$$\Pr(x_1^{1\ldots T} | \mathbf{W}_1) = \prod_{t=1}^{T_N} \Pr(\mathbf{x}_1^t | \mathbf{W}_1) \cong \prod_{t=1}^{T_N} p(\mathbf{s}_1^t | \Theta_1) | \det \mathbf{W}_1 |, \tag{6}$$

where, for convenience, $T_N = T - N + 1$. The objective function to be maximized is the multiplication of the data likelihoods of both sound sources, and we denote its log by $\mathcal{L}$:

$$
\begin{aligned}
\mathcal{L} &= \log \Pr(x_1^{1\ldots T} | \mathbf{W}_1) \Pr(x_2^{1\ldots T} | \mathbf{W}_2) \\
&\cong \sum_{t=1}^{T_N} \left[ \log p(\mathbf{s}_1^t | \Theta_1) + \log p(\mathbf{s}_2^t | \Theta_2) \right] \\
&\quad + T_N \log | \det \mathbf{W}_1 | | \det \mathbf{W}_2 |.
\end{aligned}
\tag{7}
$$

Our interest is in adapting $x_1^t$ and $x_2^t$ for $\forall t \in [1, T]$, toward the maximum of $\mathcal{L}$. We introduce a new variable $z_i^t = \lambda_i x_i^t$, a scaled value of $x_i^t$ with the contribution factor. The adaptation is done on the values of $z_i^t$ instead of $x_i^t$, in order to infer the sound sources and their contribution factors simultaneously. The learning rule is derived in a gradient-ascent manner by summing up the gradients of all the segments where the sample lies:

$$
\begin{aligned}
\frac{\partial \mathcal{L}}{\partial z_1^t} &= \sum_{n=1}^{N} \left[ \frac{\partial}{\partial z_1^t} \log p(\mathbf{s}_1^{t_n} | \Theta_1) + \frac{\partial}{\partial z_1^t} \log p(\mathbf{s}_2^{t_n} | \Theta_2) \right] \\
&= \sum_{n=1}^{N} \left[ \sum_{k=1}^{N} \left\{ \varphi(s_{1k}^{t_n}) \frac{w_{1kn}}{\lambda_1} \right\} - \sum_{k=1}^{N} \left\{ \varphi(s_{2k}^{t_n}) \frac{w_{2kn}}{\lambda_2} \right\} \right] \\
&\propto \sum_{n=1}^{N} \left[ \lambda_2 \sum_{k=1}^{N} \varphi(s_{1k}^{t_n}) w_{1kn} - \lambda_1 \sum_{k=1}^{N} \varphi(s_{2k}^{t_n}) w_{2kn} \right],
\end{aligned}
\tag{8}
$$

which is derived by the fact that $\frac{\partial s_k^{t_n}}{\partial z^t} = \frac{\partial (\mathbf{w}_k \mathbf{x}^{t_n})}{\partial x^t} \frac{\partial x^t}{\partial z^t} = \frac{w_{kn}}{\lambda}$ and $\frac{\partial z_2}{\partial z_1} = \frac{\partial (y - z_1)}{\partial z_1} = -1$, where $t_n = t - n + 1$, $\varphi(s) = \frac{\partial \log p(s|\theta)}{\partial s}$, and $w_{ikn} = \mathbf{W}_i(k, n)$. Note that the gradient of $\mathcal{L}$ for $z_2$, $\partial \mathcal{L}/\partial z_2 = -\partial \mathcal{L}/\partial z_1$, always makes the condition $y = z_1 + z_2$ satisfy, so learning rule on either $z_1$ or $z_2$ subsumes the counterpart. The overall process of the proposed method is summarized as 4 steps in Figure 2. The figure shows one iteration of the adaptation of each sample.

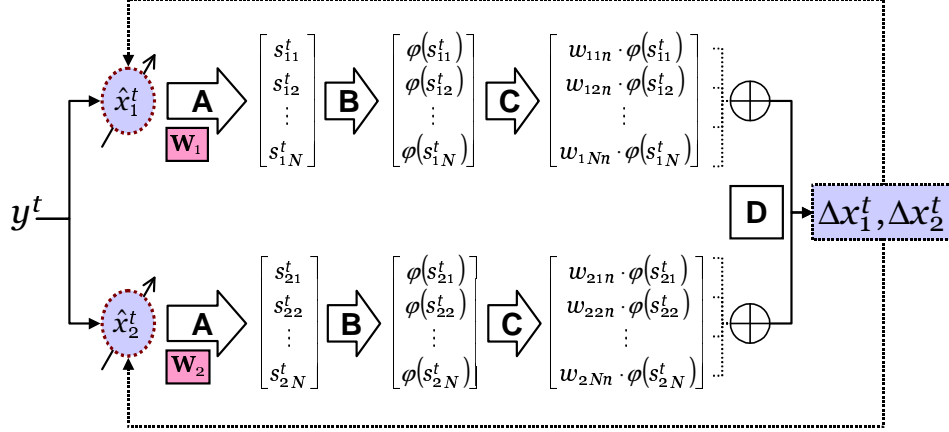

Figure 2: The overall structure of the proposed method. We are given single channel data $\mathbf{y}_t$, and we have the estimates of the source signals, $\hat{x}_i^t$, at every adaptation step. **(A)** $x_i^t \Rightarrow s_{ik}^t$: At each timepoint, the current estimates of the source signals are passed through basis filters $\mathbf{W}_i$, generating $N$ sparse codes $s_{ik}^t$ that are statistically independent. **(B)** $s_{ik}^t \Rightarrow \Delta s_{ik}^t$: The stochastic gradient for each code is obtained by taking derivative of log-likelihood. **(C)** $\Delta s_{ik}^t \Rightarrow \Delta x_i^t$: The gradient is transformed to the source domain. **(D)** The individual gradients are combined to be added to the current estimates of the source signals.

## 3.2 Estimating $\lambda_1$ and $\lambda_2$

Updating the contribution factors $\lambda_i$ can be accomplished by simply finding the maximum *a posteriori* values. To simplify the inferring steps, we force the sum of the factors to be constant: e.g. $\lambda_1 + \lambda_2 = 1$. Then $\lambda_2$ is completely dependent on $\lambda_1$ as $\lambda_2 = 1 - \lambda_1$, and we need to consider $\lambda_1$ only. Given the basis functions $\mathbf{W}_i$ and the current estimate of the sources $x_i^{1\ldots T}$, the posterior probability of $\lambda_1$ is

$$\Pr(\lambda_1 | x_1^{1\ldots T}, x_2^{1\ldots T}, \mathbf{W}_1, \mathbf{W}_2) \;\propto\; \Pr(x_1^{1\ldots T} | \mathbf{W}_1) \Pr(x_2^{1\ldots T} | \mathbf{W}_2) p_\lambda(\lambda_1) , \qquad (9)$$

where $p_\lambda(\cdot)$ is the prior density function of $\lambda_1$. The value of $\lambda_1$ maximizing the posterior probability also maximizes its log,

$$\lambda_1^* = \arg \max_{\lambda_1} \{ \mathcal{L} + \log p_\lambda(\lambda_1) \} , \qquad (10)$$

where $\mathcal{L}$ is the log-likelihood of the estimated sources defined in Equation 7. Assuming that $\lambda_1$ is uniformly distributed, $\partial \{ \mathcal{L} + \log p_\lambda(\lambda_1) \} / \partial \lambda_1 = \partial \mathcal{L} / \partial \lambda_1$, which is calculated as

$$\frac{\partial \mathcal{L}}{\partial \lambda_1} = -\frac{\psi_1}{\lambda_1^2} + \frac{\psi_2}{\lambda_2^2} , \quad \text{where} \quad \psi_i = \sum_{t=1}^{T_N} \sum_{k=1}^{N} \varphi(s_{ik}^t) \mathbf{w}_{ik} \mathbf{z}_i^t \qquad (11)$$

derived by the chain rule

$$\frac{\partial \log p(s_{ik}^t)}{\partial \lambda_i} = \frac{\partial \log p(s_{ik}^t)}{\partial s_{ik}^t} \frac{\partial s_{ik}^t}{\partial \lambda_i} = \varphi(s_{ik}^t) \cdot \mathbf{w}_{ik} \mathbf{z}_i^t \left( -\frac{1}{\lambda_i^2} \right) . \qquad (12)$$

Solving equation $\partial \mathcal{L} / \partial \lambda_1 = 0$ subject to $\lambda_1 + \lambda_2 = 1$ and $\lambda_1, \lambda_2 \in [0, 1]$ gives

$$\lambda_1^* = \frac{\sqrt{|\psi_1|}}{\sqrt{|\psi_1|} + \sqrt{|\psi_2|}} , \quad \lambda_2^* = \frac{\sqrt{|\psi_2|}}{\sqrt{|\psi_1|} + \sqrt{|\psi_2|}} . \qquad (13)$$

These values guarantee the local maxima of $\mathcal{L}$ w.r.t. the current estimates of source signals. The algorithm updates the contribution factors periodically during the learning steps.

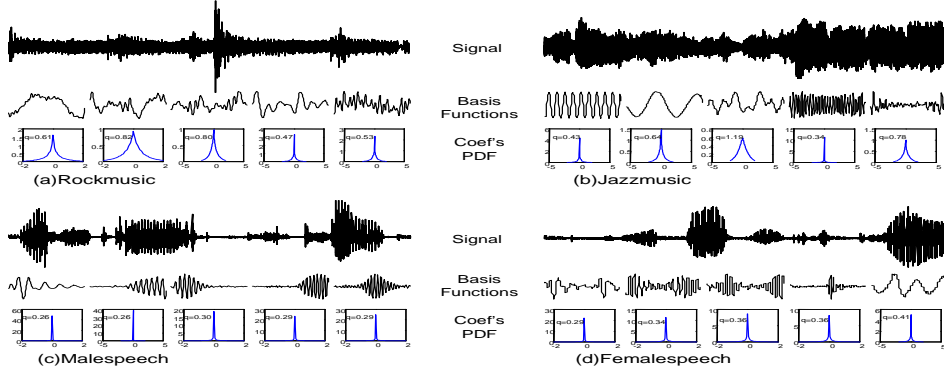

Figure 3: Waveforms of four sound sources, examples of the learned basis functions (5 were chosen out of 64), and the corresponding coefficient distributions modeled by generalized Gaussians. The full set of basis functions is available at the website also.

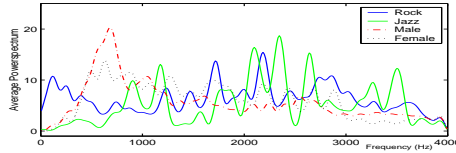

Figure 4: Average powerspectra of the 4 sound sources. Frequency scale ranges in $0 \sim 4$kHz ($x$-axis), since all the signals are sampled at 8kHz. The powerspectra are averaged and represented in the $y$-axis.

## 4  Experiments and Discussion

We have tested the performance of the proposed method on the single channel mixtures of four different sound types. They were monaural signals of rock and jazz music, male and female speech. We used different sets of speech signals for learning basis functions and for generating the mixtures. For the mixture generation, two sentences of the target speakers 'mcpm0' and 'fdaw0', one for each, were selected from the TIMIT speech database. The training set consisted of 21 sentences for each gender, 3 for each of randomly chosen 7 males (or females) from the same database excluding the 2 target speakers. Rock music was mainly composed of guitar and drum sounds, and jazz was generated by a wind instrument. Vocal parts of both music sounds were excluded. All signals were downsampled to 8kHz, from original 44.1kHz (music) and 16kHz (speech) data. The training data were segmented in 64 samples (8ms) starting at every sample. Audio files for all the experiments are accessible at the website[1].

Figure 3 displays the actual sources, adapted basis functions, and their coefficient distributions. Music basis functions exhibit consistent amplitudes with harmonics, and the speech basis functions are similar to Gabor wavelets. Figure 4 compares 4 sources by the average spectra. Each covers all the frequency bands, although they are different in amplitude. One might expect that simple filtering or masking cannot separate the mixed sources clearly.

Before actual separation, the source signals were initialized to the values of mixture signal: $x_i^t = y^t$, and the initial $\lambda_i$ were all 0.5 to satisfy Equation 1. The adaptation was repeated more than 300 steps on each sample, and the scaling factors were updated every 10 steps. Table 1 reports the signal-to-noise ratios (SNRs) of the mixed signal ($y^t$) and the recovered results ($\hat{z}_i^t$) with the original sources ($z_i^t = \lambda_i x_i^t$). In terms of total SNR increase the mixtures containing music were recovered more cleanly than the male-female mixture. Separation of jazz music and male speech was the best, and the waveforms are illustrated

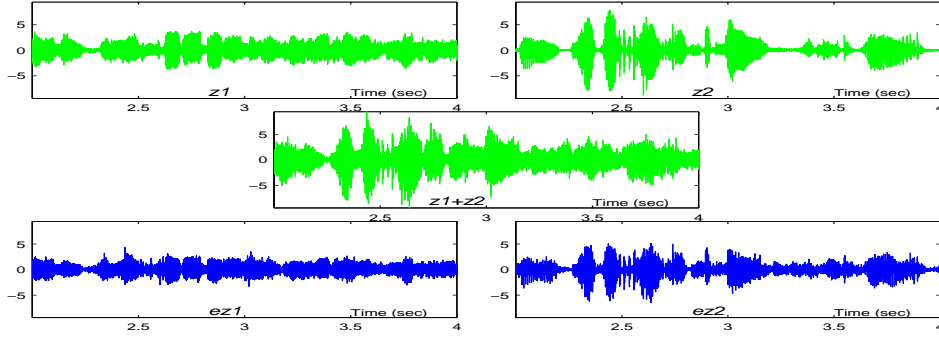

Figure 5: Separation result for the mixture of jazz music and male speech. In the vertical order: original sources ($z_1$ and $z_2$), mixed signal ($z_1 + z_2$), and the recovered signals.

in Figure 5. We conjecture by the average spectra of the sources in Figure 4 that although there exists plenty of overlap between jazz and speech, the structures are dissimilar, i.e. the frequency components of jazz change less, so we were able to obtain relatively good SNR results. However rock music exhibits scattered spectrum and less characteristical structure. This explains the relatively poorer performances of rock mixtures.

It is very difficult to compare a separation method with other CASA techniques, because their approaches are so different in many ways that an optimal tuning of their parameters would be beyond the scope of this paper. However, we compared our method with Wiener filtering [4], that provides optimal masking filters in the frequency domain if true spectrogram is given. So, we assumed that the other source was completely known. The filters were computed every block of 8 ms (64 samples), 0.5 sec, and 1.0 sec. In this case, our blind results were comparable in SNR with results obtained when the Wiener filters were computed at 0.5 sec.

In summary, our method has several advantages over traditional approaches to signal separation. They involve either spectral techniques [5, 6] or time-domain nonlinear filtering techniques [3, 4]. Spectral techniques assume that sources are disjoint in the spectrogram, which frequently result in audible distortions of the signal in the region where the assumption mismatches. Recent time-domain filtering techniques are based on splitting the whole signal space into several disjoint subspaces. Although they overcome the limit of spectral representation, they consider second-order statistics only, such as autocorrelation, which restricts the separable cases to orthogonal subspaces [4].

Our method avoids these strong assumptions by utilizing a prior set of basis functions that captures the inherent statistical structures of the source signal. This generative model therefore makes use of spectral and temporal structures at the same time. The constraints are dictated by the ICA algorithm that forces the basis functions to result in an efficient representation, i.e. the linearly independent source coefficients; and both, the basis functions

Table 1: SNR results. {R, J, M, F} stand for rock, jazz music, male, and female speech. All the values are measured in dB. *'Mix'* columns are the sources that are mixed to $y$, and 'snr$_{z_i}$'s are the calculated SNR of mixed signal ($y$) and recovered sources ($\hat{z}_i$) with the original sources ($z_i = \lambda_i x_i$).

| *Mix* | snr$_{s_1}$ | | snr$_{s_2}$ | | *Total* | *Mix* | snr$_{s_1}$ | | snr$_{s_2}$ | | *Total* |
|---|---|---|---|---|---|---|---|---|---|---|---|
| | $m$ | $y_1$ | $m$ | $y_2$ | *inc.* | | $m$ | $y_1$ | $m$ | $y_2$ | *inc.* |
| R + J | -3.7 | 3.3 | 3.7 | 7.0 | 10.3 | J + M | 0.1 | 5.6 | -0.1 | 5.5 | **11.1** |
| R + M | -3.7 | 3.1 | 3.7 | 6.8 | 9.9 | J + F | -0.1 | 5.1 | 0.1 | 5.3 | 10.4 |
| R + F | -3.9 | 2.2 | 3.9 | 6.1 | 8.3 | M + F | -0.2 | 2.5 | 0.2 | 2.7 | 5.2 |

and their corresponding pdf are key to obtaining a faithful MAP based inference algorithm. An important question is how well the traing data has to match the test data. We have also performed experiments with the set of basis functions learned from the test sounds and the SNR decreased on average by 1dB.

## 5 Conclusions

We presented a technique for single channel source separation utilizing the time-domain ICA basis functions. Instead of traditional prior knowledge of the sources, we exploited the statistical structures of the sources that are inherently captured by the basis and its coefficients from a training set. The algorithm recovers original sound streams through gradient-ascent adaptation steps pursuing the maximum likelihood estimate, contraint by the parameters of the basis filters and the generalized Gaussian distributions of the filter coefficients. With the separation results, we demonstrated that the proposed method is applicable to the real world problems such as blind source separation, denoising, and restoration of corrupted or lost data. Our current research includes the extension of this framework to perform model comparision to estimate which set of basis functions to use given a dictionary of basis functions. This is achieved by applying a variational Bayes method to compare different basis function models to select the most likely source. This method also allows us to cope with other unknown parameters such the as the number of sources. Future work will address the optimization of the learning rules towards real-time processing and the evaluation of this methodology with speech recognition tasks in noisy environments, such as the AURORA database.

## Footnotes

[1] http://speech.kaist.ac.kr/~jangbal/ch1bss/

## References

[1] G. J. Brown and M. Cooke, "Computational auditory scene analysis," *Computer Speech and Language*, vol. 8, no. 4, pp. 297–336, 1994.

[2] P. Comon, "Independent component analysis, A new concept?," *Signal Processing*, vol. 36, pp. 287–314, 1994.

[3] E. Wan and A. T. Nelson, "Neural dual extended kalman filtering: Applications in speech enhancement and monaural blind signal separation," in *Proc. of IEEE Workshop on Neural Networks and Signal Processing*, 1997.

[4] J. Hopgood and P. Rayner, "Single channel signal separation using linear time-varying filters: Separability of non-stationary stochastic signals," in *Proc. ICASSP*, vol. 3, (Phoenix, Arizona), pp. 1449–1452, March 1999.

[5] S. T. Roweis, "One microphone source separation," *Advances in Neural Information Processing Systems*, vol. 13, pp. 793–799, 2001.

[6] S. Rickard, R. Balan, and J. Rosca, "Real-time time-frequency based blind source separation," in *Proc. of International Conference on Independent Component Analysis and Signal Separation (ICA2001)*, (San Diego, CA), pp. 651–656, December 2001.

[7] T.-W. Lee and G.-J. Jang, "The statistical structures of male and female speech signals," in *Proc. ICASSP*, (Salt Lake City, Utah), May 2001.

[8] A. J. Bell and T. J. Sejnowski, "Learning the higher-order structures of a natural sound," *Network: Computation in Neural Systems*, vol. 7, pp. 261–266, July 1996.

[9] T.-W. Lee and M. S. Lewicki, "The generalized Gaussian mixture model using ICA," in *International Workshop on Independent Component Analysis (ICA'00)*, (Helsinki, Finland), pp. 239–244, June 2000.

[10] B. Pearlmutter and L. Parra, "A context-sensitive generalization of ICA," in *Proc. ICONIP*, (Hong Kong), pp. 151–157, September 1996.
